# A Conditional Multinomial Mixture Model for Superset Label Learning

**Li-Ping Liu**
EECS, Oregon State University
Corvallis, OR 97331
liuli@eecs.oregonstate.edu

**Thomas G. Dietterich**
EECS, Oregon State University
Corvallis, OR 97331
tgd@cs.orst.edu

## Abstract

In the superset label learning problem (SLL), each training instance provides a *set* of candidate labels of which one is the true label of the instance. As in ordinary regression, the candidate label set is a noisy version of the true label. In this work, we solve the problem by maximizing the likelihood of the candidate label sets of training instances. We propose a probabilistic model, the Logistic Stick-Breaking Conditional Multinomial Model (LSB-CMM), to do the job. The LSB-CMM is derived from the logistic stick-breaking process. It first maps data points to mixture components and then assigns to each mixture component a label drawn from a component-specific multinomial distribution. The mixture components can capture underlying structure in the data, which is very useful when the model is weakly supervised. This advantage comes at little cost, since the model introduces few additional parameters. Experimental tests on several real-world problems with superset labels show results that are competitive or superior to the state of the art. The discovered underlying structures also provide improved explanations of the classification predictions.

## 1 Introduction

In supervised classification, the goal is to learn a classifier from a collection of training instances, where each instance has a unique class label. However, in many settings, it is difficult to obtain such precisely-labeled data. Fortunately, it is often possible to obtain a *set* of labels for each instance, where the correct label is one of the elements of the set.

For example, captions on pictures (in newspapers, facebook, etc.) typically identify all of the people the picture but do not necessarily indicate which face belongs to each person. Imprecisely-labeled training examples can be created by detecting each face in the image and defining a label set containing all of the names mentioned in the caption. A similar case arises in bird song classification [2]. In this task, a field recording of multiple birds singing is divided into 10-second segments, and experts identify the species of all of the birds singing in each segment without localizing each species to a specific part of the spectrogram. These examples show that superset-labeled data are typically much cheaper to acquire than standard single-labeled data. If effective learning algorithms can be devised for superset-labeled data, then they would have wide application.

The superset label learning problem has been studied under two main formulations. In the multi-instance multi-label (MIML) formulation [15], the training data consist of pairs $(B_i, Y_i)$, where $B_i = \{\mathbf{x}_{i,1}, \ldots, \mathbf{x}_{i,n_i}\}$ is a set of instances and $Y_i$ is a set of labels. The assumption is that for every instance $\mathbf{x}_{i,j} \in B_i$, its true label $y_{i,j} \in Y_i$. The work of Jie et al. [9] and Briggs et al. [2] learn classifiers from such set-labeled bags.

In the superset label formulation (which has sometimes been confusingly called the "partial label" problem) [7, 10, 8, 12, 4, 5], each instance $\mathbf{x}_n$ has a candidate label set $Y_n$ that contains the unknown

true label $y_n$. This formulation ignores any bag structure and views each instance independently. It is more general than the MIML formulation, since any MIML problem can be converted to a super-set label problem (with loss of the bag information). Furthermore, the superset label formulation is natural in many applications that do not involve bags of instances. For example, in some applications, annotators may be unsure of the correct label, so permitting them to provide a superset of the correct label avoids the risk of mislabeling. In this paper, we employ the superset label formulation. Other relevant work includes Nguyen et al. [12] and Cour et al. [5] who extend SVMs to handle superset labeled data.

In the superset label problem, the label set $Y_n$ can be viewed as a corruption of the true label. The standard approach to learning with corrupted labels is to assume a generic noise process and incorporate it into the likelihood function. In standard supervised learning, it is common to assume that the observed label is sampled from a Bernoulli random variable whose most likely outcome is equal to the true label. In ordinary least-squares regression, the assumption is that the observed value is drawn from a Gaussian distribution whose mean is equal to the true value and whose variance is a constant $\sigma^2$. In the superset label problem, we will assume that the observed label set $Y_n$ is drawn from a set-valued distribution $p(Y_n|y_n)$ that depends only on the true label. When computing the likelihood, this will allow us to treat the true label as a latent variable that can be marginalized away.

When the label information is imprecise, the learning algorithm has to depend more on underlying structure in the data. Indeed, many semi-supervised learning methods [16] model cluster structure of the training data explicitly or implicitly. This suggests that the underlying structure of the data should also play important role in the superset label problem.

In this paper, we propose the Logistic Stick-Breaking Conditional Multinomial Model (LSB-CMM) for the superset label learning problem. The model has two components: the mapping component and the coding component. Given an input $\mathbf{x}_n$, the mapping component maps $\mathbf{x}_n$ to a region $k$. Then the coding component generates the label according to a multinomial distribution associated with $k$. The mapping component is implemented by the Logistic Stick Breaking Process(LSBP) [13] whose Bernoulli probabilities are from discriminative functions. The mapping and coding components are optimized simultaneously with the variational EM algorithm.

LSB-CMM addresses the superset label problem in several aspects. First, the mapping component models the cluster structure with a set of regions. The fact that instances in the same region often have the same label is important for inferring the true label from noisy candidate label sets. Second, the regions do not directly correspond to classes. Instead, the number of regions is automatically determined by data, and it can be much larger than the number of classes. Third, the results of the LSB-CMM model can be more easily interpreted than the approaches based on SVMs [5, 2]. The regions provide information about how data are organized in the classification problem.

## 2 The Logistic Stick Breaking Conditional Multinomial Model

The superset label learning problem seeks to train a classifier $f : \mathcal{R}^d \mapsto \{1, \cdots, L\}$ on a given dataset $(\mathbf{x}, Y) = \{(\mathbf{x}_n, Y_n)\}_{n=1}^N$, where each instance $\mathbf{x}_n \in \mathcal{R}^d$ has a *candidate label set* $Y_n \subset \{1, \cdots, L\}$. The true labels $y = \{y_n\}_{n=1}^N$ are not directly observed. The only information is that the true label $y_n$ of instance $\mathbf{x}_n$ is in the candidate set $Y_n$. The extra labels $\{l|l \neq y_n, l \in Y_n\}$ causing ambiguity will be called the *distractor labels*. For any test instance $(\mathbf{x}_t, y_t)$ drawn from the same distribution as $\{(\mathbf{x}_n, y_n)\}_{n=1}^N$, the trained classifier $f$ should be able to map $\mathbf{x}_t$ to $y_t$ with high probability. When $|Y_n| = 1$ for all $n$, the problem is a supervised classification problem. We require $|Y_n| < L$ for all $n$; that is, every candidate label set must provide at least some information about the true label of the instance.

### 2.1 The Model

As stated in the introduction, the candidate label set is a noisy version of the true label. To train a classifier, we first need a likelihood function $p(Y_n|\mathbf{x}_n)$. The key to our approach is to write this as $p(Y_n|\mathbf{x}_n) = \sum_{y_n=1}^L p(Y_n|y_n)p(y_n|\mathbf{x}_n)$, where each term is the product of the underlying true classifier, $p(y_n|\mathbf{x}_n)$, and the noise model $p(Y_n|y_n)$. We then make the following assumption about the noise distribution:

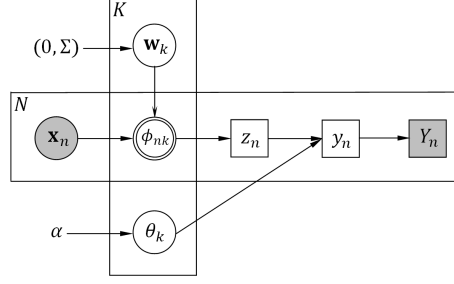

Figure 1: The LSB-CMM. Square nodes are discrete, circle nodes are continuous, and double-circle nodes are deterministic.

**Assumption:** *All labels in the candidate label set $Y_n$ have the same probability of generating $Y_n$, but no label outside of $Y_n$ can generate $Y_n$*

$$p(Y_n|y_n = l) = \begin{cases} \lambda(Y_n) & \text{if } l \in Y_n \\ 0 & \text{if } l \notin Y_n \end{cases} . \qquad (1)$$

This assumption enforces three constraints. First, the set of labels $Y_n$ is conditionally independent of the input $\mathbf{x}_n$ given $y_n$. Second, labels that do not appear in $Y_n$ have probability 0 of generating $Y_n$. Third, all of the labels in $Y_n$ have equal probability of generating $Y_n$ (symmetry). Note that these constraints do *not* imply that the training data are correctly labeled. That is, suppose that the most likely label for a particular input $\mathbf{x}_n$ is $y_n = l$. Because $p(y_n|\mathbf{x}_n)$ is a multinomial distribution, a different label $y_n = l'$ might be assigned to $\mathbf{x}_n$ by the labeling process. Then this label is further corrupted by adding distractor labels to produce $Y_n$. Hence, it could be that $l \notin Y_n$. In short, in this model, we have the usual "multinomial noise" in the labels which is then further compounded by "superset noise". The third constraint can be criticized for being simplistic; we believe it can be replaced with a learned noise model in future work.

Given (1), we can marginalize away $y_n$ in the following optimization problem maximizing the likelihood of observed candidate labels.

$$\begin{aligned} f^* &= \arg\max_f \sum_{n=1}^{N} \log \sum_{y_n=1}^{L} p(y_n|\mathbf{x}_n; f) p(Y_n|y_n) \\ &= \arg\max_f \sum_{n=1}^{N} \log \sum_{y_n \in Y_n} p(y_n|\mathbf{x}_n; f) + \sum_{n=1}^{N} \log(\lambda(Y_n)). \end{aligned} \qquad (2)$$

Under the conditional independence and symmetry assumptions, the last term does not depend on $f$ and so can be ignored in the optimization. This result is consistent with the formulation in [10].

We propose the Logistic Stick-Breaking Conditional Multinomial Model to instantiate $f$ (see Figure 1). In LSB-CMM, we introduce a set of $K$ regions (mixture components) $\{1, \ldots, K\}$. LSB-CMM has two components. The mapping component maps each instance $\mathbf{x}_n$ to a region $z_n, z_n \in \{1, \ldots, K\}$. Then the coding component draws a label $y_n$ from the multinomial distribution indexed by $z_n$ with parameter $\theta_{z_n}$. We denote the region indexes of the training instances by $z = (z_n)_{n=1}^{N}$.

In the mapping component, we employ the Logistic Stick Breaking Process(LSBP) [13] to model the instance-region relationship. LSBP is a modification of the Dirichlet Process (DP) [14]. In LSBP, the sequence of Bernoulli probabilities are the outputs of a sequence of logistic functions instead of being random draws from a Beta distribution as in the Dirichlet process. The input to the $k$-th logistic function is the dot product of $\mathbf{x}_n$ and a learned weight vector $\mathbf{w}_k \in R^{d+1}$. (The added dimension corresponds to a zeroth feature fixed to be 1 to provide an intercept term.) To regularize these logistic functions, we posit that each $\mathbf{w}_k$ is drawn from a Gaussian distribution Normal$(0, \Sigma)$, where $\Sigma = \text{diag}(\infty, \sigma^2, \cdots, \sigma^2)$. This regularizes all terms in $\mathbf{w}_k$ except the intercept. For each $\mathbf{x}_n$, a sequence of probabilities $\{v_{nk}\}_{k=1}^{K}$ is generated from logistic functions, where $v_{nk} = \text{expit}(\mathbf{w}_k^T \mathbf{x}_n)$ and $\text{expit}(u) = 1/(1 + \exp(-u))$ is the logistic function. We truncate $k$ at $K$ by setting $\mathbf{w}_K = (+\infty, 0, \cdots, 0)$ and thus $v_{nK} = 1$. Let $\mathbf{w}$ denote the collection of all $K$ $\mathbf{w}_k$. Given the probabilities

$v_{n1}, \ldots, v_{nK}$ computed from $\mathbf{x}_n$, we choose the region $z_n$ according to a stick-breaking procedure:

$$p(z_n = k) = \phi_{nk} = v_{nk} \prod_{i=1}^{k-1} (1 - v_{ni}). \tag{3}$$

Here we stipulate that the product is 1 when $k = 1$. Let $\phi_n = (\phi_{n1}, \cdots, \phi_{nK})$ constitute the parameter of a multinomial distribution. Then $z_n$ is drawn from this distribution.

In the coding component of LSB-CMM, we first draw $K$ $L$-dimensional multinomial probabilities $\theta = \{\theta_k\}_{k=1}^{K}$ from the prior Dirichlet distribution with parameter $\alpha$. Then, for each instance $\mathbf{x}_n$ with mixture $z_n$, its label $y_n$ is drawn from the multinomial distribution with $\theta_{z_n}$. In the traditional multi-class problem, $y_n$ is observed. However, in the SLL problem $y_n$ is not observed and $Y_n$ is generated from $y_n$.

The generative process of the whole model is summarized below:

$$\mathbf{w}_k \quad \sim \quad \text{Normal}(0, \Sigma), 1 \le k \le K - 1, \quad \mathbf{w}_K = (+\infty, 0, \cdots, 0) \tag{4}$$

$$z_n \quad \sim \quad \text{Mult}(\phi_n), \quad \phi_{nk} = \text{expit}(\mathbf{w}_k^T \mathbf{x}_n) \prod_{i=1}^{k-1} (1 - \text{expit}(\mathbf{w}_i^T \mathbf{x}_n)) \tag{5}$$

$$\theta_k \quad \sim \quad \text{Dirichlet}(\alpha) \tag{6}$$

$$y_n \quad \sim \quad \text{Mult}(\theta_{z_n}) \tag{7}$$

$$Y_n \quad \sim \quad \text{Dist1}(y_n) \quad (\text{Dist1 is some distribution satisfying (1)}) \tag{8}$$

As shown in (2), the model needs to maximize the likelihood that each $y_n$ is in $Y_n$. After incorporating the priors, we can write the penalized maximum likelihood objective as

$$\max LL = \sum_{n=1}^{N} \log \left( \sum_{y_n \in Y_n} p(y_n | \mathbf{x}_n, \mathbf{w}, \alpha) \right) + \log(p(\mathbf{w}|0, \Sigma)). \tag{9}$$

This cannot be solved directly, so we apply variational EM [1].

## 2.2 Variational EM

The hidden variables in the model are $y$, $z$, and $\theta$. For these hidden variables, we introduce the variational distribution $q(y, z, \theta | \hat{\phi}, \hat{\alpha})$, where $\hat{\phi} = \{\hat{\phi}_n\}_{n=1}^{N}$ and $\hat{\alpha} = \{\hat{\alpha}_k\}_{k=1}^{K}$ are the parameters. Then we factorize $q$ as

$$q(z, y, \theta | \hat{\phi}, \hat{\alpha}) = \prod_{n=1}^{N} q(z_n, y_n | \hat{\phi}_n) \prod_{k=1}^{K} q(\theta_k | \hat{\alpha}_k), \tag{10}$$

where $\hat{\phi}_n$ is a $K \times L$ matrix and $q(z_n, y_n | \hat{\phi}_n)$ is a multinomial distribution in which $p(z_n = k, y_n = l) = \hat{\phi}_{nkl}$. This distribution is constrained by the candidate label set: if a label $l \notin Y_n$, then $\hat{\phi}_{nkl} = 0$ for any value of $k$. The distribution $q(\theta_k | \hat{\alpha}_k)$ is a Dirichlet distribution with parameter $\hat{\alpha}_k$.

After we set the distribution $q(z, y, \theta)$, our variational EM follows standard methods. The detailed derivation can be found in the supplementary materials [11]. Here we only show the final updating step with some analysis.

In the E step, the parameters of variational distribution are updated as (11) and (12).

$$\hat{\phi}_{nkl} \quad \propto \quad \begin{cases} \phi_{nk} \exp\left(E_{q(\theta_k | \hat{\alpha}_k)}[\log(\theta_{kl})]\right), & \text{if } l \in Y_n \\ 0, & \text{if } l \notin Y_n \end{cases}, \tag{11}$$

$$\hat{\alpha}_k \quad = \quad \alpha + \sum_{n=1}^{N} \hat{\phi}_{nkl}. \tag{12}$$

The update of $\hat{\phi}_n$ in (11) indicates the key difference between the LSB-CMM model and traditional clustering models. The formation of regions is directed by both instance similarities and class labels.

If the instance $\mathbf{x}_n$ wants to join region $k$ (i.e., $\sum_l \hat{\phi}_{nkl}$ is large), then it must be similar to $\mathbf{w}_k$ as well as to instances in that region in order to make $\phi_{nk}$ large. Simultaneously, its candidate labels must fit the "label flavor" of region $k$, where the "label flavor" means region $k$ prefers labels having large values in $\hat{\alpha}_k$. The update of $\hat{\alpha}$ in (12) can be interpreted as having each instance $\mathbf{x}_n$ vote for the label $l$ for region $k$ with weight $\hat{\phi}_{nkl}$.

In the M step, we need to solve the maximization problem in (13) for each $\mathbf{w}_k, 1 \leq k \leq K - 1$. Note that $\mathbf{w}_K$ is fixed. Each $\mathbf{w}_k$ can be optimized separately. The optimization problem is similar to the problem of logistic regression and is also a concave maximization problem, which can be solved by any gradient-based method, such as BFGS.

$$\max_{\mathbf{w}_k} \quad -\frac{1}{2}\mathbf{w}_k^T \Sigma^{-1} \mathbf{w}_k + \sum_{n=1}^{N} \left[ \hat{\phi}_{nk} \log(\text{expit}(\mathbf{w}_k^T \mathbf{x}_n)) + \hat{\psi}_{nk} \log(1 - \text{expit}(\mathbf{w}_k^T \mathbf{x}_n)) \right], \qquad (13)$$

where $\hat{\phi}_{nk} = \sum_{l=1}^{L} \hat{\phi}_{nkl}$ and $\hat{\psi}_{nk} = \sum_{j=k+1}^{K} \hat{\phi}_{nj}$. Intuitively, the variable $\hat{\phi}_{nk}$ is the probability that instance $\mathbf{x}_n$ belongs to region $k$, and $\hat{\psi}_{nk}$ is the probability that $\mathbf{x}_n$ belongs to region $\{k+1, \cdots, K\}$. Therefore, the optimal $\mathbf{w}_k$ discriminates instances in region $k$ against instances in regions $\geq k$.

## 2.3 Prediction

For a test instance $\mathbf{x}_t$, we predict the label with maximum posterior probability. The test instance can be mapped to a region with $\mathbf{w}$, but the coding matrix $\theta$ is marginalized out in the EM. We use the variational distribution $p(\theta_k | \hat{\alpha}_k)$ as the prior of each $\theta_k$ and integrate out all $\theta_k$-s. Given a test point $\mathbf{x}_t$, the prediction is the label $l$ that maximizes the probability $p(y_t = l | \mathbf{x}_t, \mathbf{w}, \hat{\alpha})$ calculated as (14). The detailed derivation is also in the supplementary materials [11].

$$p(y_t = l | \mathbf{x}_t, \mathbf{w}, \hat{\alpha}) \quad = \quad \sum_{k=1}^{K} \phi_{tk} \frac{\hat{\alpha}_{kl}}{\sum_l \hat{\alpha}_{kl}} , \qquad (14)$$

where $\phi_{tk} = \left( \text{expit}(\mathbf{w}_k^T \mathbf{x}_t) \prod_{i=1}^{k-1} (1 - \text{expit}(\mathbf{w}_i^T \mathbf{x}_t)) \right)$. The test instance goes to region $k$ with probability $\phi_{tk}$, and its label is decided by the votes ($\hat{\alpha}_k$) in that region.

## 2.4 Complexity Analysis and Practical Issues

In the E step, for each region $k$, the algorithm iterates over all candidate labels of all instances, so the complexity is $O(NKL)$. In the $M$ step, the algorithm solves $K-1$ separate optimization problems. Suppose each optimization problem takes $O(VNd)$ time, where $V$ is the number of BFGS iterations. Then the complexity is $O(KVNd)$. Since $V$ is usually larger than $L$, the overall complexity of one EM iteration is $O(KVNd)$. Suppose the EM steps converge within $m$ iterations, where $m$ is usually less than 50. Then the overall complexity is $O(mKVNd)$. The space complexity is $O(NK)$, since we only store the matrix $\sum_{l=1}^{L} \hat{\phi}_{nkl}$ and the matrix $\hat{\alpha}$.

In prediction, the mapping phase requires $O(Kd)$ time to multiply $\mathbf{w}$ and the test instance. After the stick breaking process, which takes $O(K)$ calculations, the coding phase requires $O(KL)$ calculation. Thus the overall time complexity is $O(K \max\{d, L\})$. Hence, the prediction time is comparable to that of logistic regression.

There are several practical issues that affect the performance of the model. **Initialization:** From the model design, we can expect that instances in the same region have the same label. Therefore, it is reasonable to initialize $\hat{\alpha}$ to have each region prefer only one label, that is, each $\hat{\alpha}_k$ has one element with large value and all others with small values. We initialize $\phi$ to $\phi_{nk} = \frac{1}{K}$, so that all regions have equal probability to be chosen at the start. Initialization of these two variables is enough to begin the EM iterations. We find that such initialization works well for our model and generally is better than random initialization. **Calculation of** $E_{q(\theta_k | \hat{\alpha}_k)}[\log(\theta_{kl})]$ **in (11):** Although it has a closed-form solution, we encountered numerical issues, so we calculate it via Monte Carlo sampling. This does not change complexity analysis above, since the training is dominated by M step. **Priors:** We found that using a non-informative prior for Dirichlet($\alpha$) worked best. From (12) and (14), we can see that when $\theta$ is marginalized, the distribution is non-informative when $\alpha$ is set to small values. We use $\alpha = 0.05$ in our experiments.

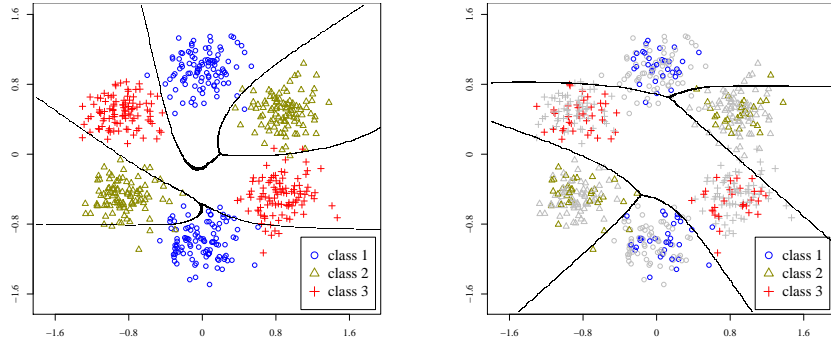

Figure 2: Decision boundaries of LSB-CMM on a linearly-inseparable problem. Left: all data points have true labels. Right: labels of gray data points are corrupted.

# 3 Experiments

In this section, we describe the results of several experiments we conducted to study the behavior of our proposed model. First, we experiment with a toy problem to show that our algorithm can solve problems with linearly-inseparable classes. Second, we perform controlled experiments on three synthetic datasets to study the robustness of LSB-CMM with respect to the degree of ambiguity of the label sets. Third, we experiment with three real-world datasets.

**LSB-CMM Model:** The LSB-CMM model has three parameters $K, \sigma^2, \alpha$. We find that the model is insensitive to $K$ if it is sufficiently large. We set $K = 10$ for the toy problems and $K = 5L$ for other problems. $\alpha$ is set to $0.05$ for all experiments. When the data is standardized, the regularization parameter $\sigma^2 = 1$ generally gives good results, so $\sigma^2$ is set to 1 in all superset label tasks.

**Baselines:** We compared the LSB-CMM model with three state-of-the-art methods. **Supervised SVM:** the SVM is always trained with the true labels. Its performance can be viewed as an upper bound on the performance of any SSL algorithm. LIBSVM [3] with RBF kernel was run to construct a multi-class classifier in *one-vs-one* mode. One third of the training data was used to tune the $C$ parameter and the RBF kernel parameter $\gamma$. **CLPL:** CLPL [5] is a linear model that encourages large average scores of candidate labels. The model is insensitive to the $C$ parameter, so we set the $C$ value to 1000 (the default value in their code). **SIM:** SIM [2] minimizes the ranking loss of instances in a bag. In controlled experiments and in one of the real-world problems, we could not make the comparison to LSB-CMM because of the lack of bag information. The $\lambda$ parameter is set to $10^{-8}$ based on authors' recommendation.

## 3.1 A Toy Problems

In this experiment, we generate a linearly-inseparable SLL problem. The data has two dimensions and six clusters drawn from six normal distributions with means at the corners of a hexagon. We assign a label to each cluster so that the problem is linearly-inseparable (see (2)). In the first task, we give the model the true labels. In the second task, we add a distractor label for two thirds of all instances (gray data points in the figure). The distractor label is randomly chosen from the two labels other than the true label. The decision boundaries found by LSB-CMM in both tasks are shown in (2)). We can see that LSB-CMM can successfully give nonlinear decision boundaries for this problem. After injecting distractor labels, LSB-CMM still recovers the boundaries between classes. There is minor change of the boundary at the edge of the cluster, while the main part of each cluster is classified correctly.

## 3.2 Controlled Experiments

We conducted controlled experiments on three UCI [6] datasets: $\{segment$ (2310 instances, 7 classes), $pendigits$ (10992 instances, 10 classes), and $usps$ (9298 instances, 10 classes)$\}$. Ten-fold cross validation is performed on all three datasets. For each training instance, we add distractor labels with controlled probability. As in [5], we use $p, q,$ and $\varepsilon$ to control the ambiguity level of

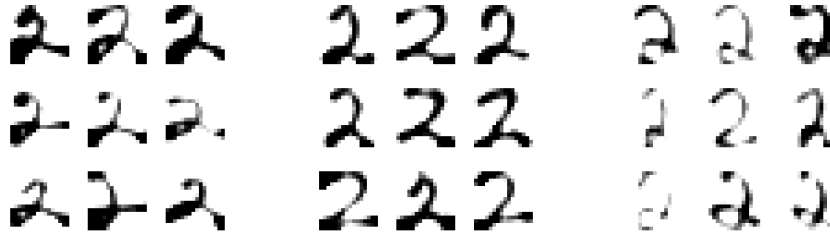

Figure 3: Three regions learned by the model on $usps$

candidate label sets. The roles and values of these three variables are as follows: $p$ is the probability that an instance has distractor labels ($p = 1$ for all controlled experiments); $q \in \{1, 2, 3, 4\}$ is the number of distractor labels; and $\varepsilon \in \{0.3, 0.7, 0.9, 0.95\}$ is the maximum probability that a distractor label co-occurs with the true label [5], also called the ambiguity degree.

We have two settings for these three variables. In the first setting, we hold $q = 1$ and vary $\varepsilon$, that is, for each label $l$, we choose a specific label $l' \neq l$ as the (unique) distractor label with probability $\varepsilon$ or choose any other label with probability $1 - \varepsilon$. In the extreme case when $\varepsilon = 1$, $l'$ and $l$ always co-occur, and they cannot be distinguished by any classifier. In the second setting, we vary $q$ and pick distractor labels randomly for each candidate label set.

The results are shown in Figure (4). Our LSB-CMM model significantly outperforms the CLPL approach. As the number of distractor labels increases, performance of both methods goes down, but not too much. When the true label is combined with different distractor labels, the disambiguation is easy. The co-occurring distractor labels provide much less disambiguation. This explains why large ambiguity degree hurts the performance of both methods. The small dataset ($segment$) suffers even more from large ambiguity degree, because there are fewer data points that can "break" the strong correlation between the true label and the distractors.

To explore why the LSB-CMM model has good performance, we investigated the regions learned by the model. Recall that $\phi_{nk}$ is the probability that $\mathbf{x}_n$ is sent to region $k$. In each region $k$, the representative instances have large values of $\phi_{nk}$. We examined all $\phi_{nk}$ from the model trained on the $usps$ dataset with 3 random distractor labels. For each region $k$, we selected the 9 most representative instances. Figure (3) shows representative instances for three regions. These are all from class "2" but are written in different styles. This shows that the LSB-CMM model can discover the sub-classes in the data. In some applications, the whole class is not easy to discriminate from other classes, but sometimes each sub-class can be easily identified. In such cases, LSB-CMM will be very useful and can improve performance.

Explanation of the results via regions can also give better understanding of the learned classifier. In order to analyze the performance of the classifier learned from data with either superset labels or fully observed labels, one traditional method is to compute the confusion matrix. While the confusion matrix can only tell the relationships between classes, the mixture analysis can indicate precisely which subclass of a class are confused with which subclasses of other classes. The regions can also help the user identify and define new classes as refinements of existing ones.

### 3.3 Real-World Problems

We apply our model on three real-world problems. **1) BirdSong dataset** [2]: This contains 548 10-second bird song recordings. Each recording contains 1-40 syllables. In total there are 4998 syllables. Each syllable is described by 38 features. The labels of each recording are the bird species that were singing during that 10-second period, and these species become candidate labels set of each syllable in the recording. **2) MSRCv2 dataset**: This dataset contains 591 images with 23 classes. The ground truth segmentations (regions with labels) are given. The labels of all segmentations in an image are treated as candidate labels for each segmentation. Each segmentation is described by 48-dimensional gradient and color histograms. **3) Lost dataset** [5]: This dataset contains 1122 faces, and each face has the true label and a set of candidate labels. Each face is described by 108 PCA components. Since the bag information (i.e., which faces are in the same scene) is missing,

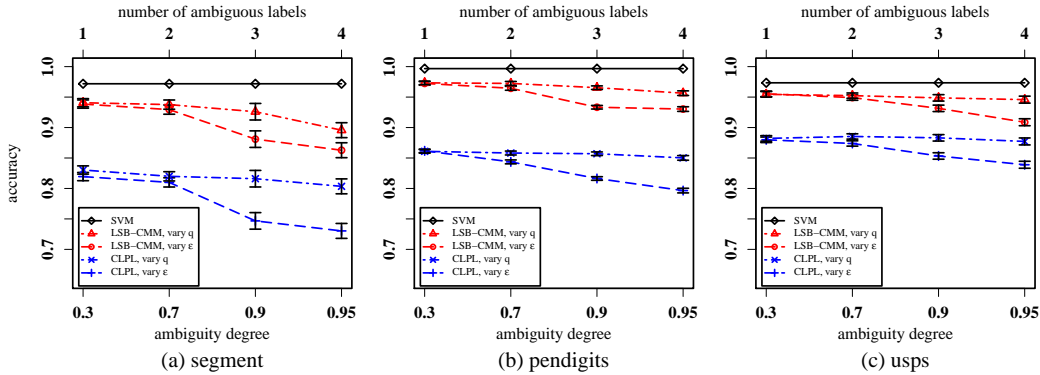

Figure 4: Classification performance on synthetic data (red: LSB-CMM; blue: CLPL). The dot-dash line is for different $q$ values (number of distractor labels) as shown on the top x-axis. The dashed line is for different $\varepsilon$ (ambiguity degree) values as shown on the bottom x-axis.

Table 1: Classification Accuracies for Superset Label Problems

|  | LSB-CMM | SIM | CLPL | SVM |
|---|---|---|---|---|
| BirdSong | **0.715(0.042**) | 0.589(0.035) | 0.637(0.034) | 0.790(0.027) |
| MSRCv2 | **0.459(0.032**) | **0.454(0.043**) | 0.411(0.044) | 0.673(0.043) |
| Lost | **0.703(0.058)** | - | **0.710(0.045)** | 0.817(0.038) |

SIM is not compared to our model on this dataset. We run 10-fold cross validation on these three datasets. The BirdSong and MSRCv2 datasets are split by recordings/images, and the Lost dataset is split by faces.

The classification accuracies are shown in Table (1). Accuracies of the three superset label learning algorithms are compared using the paired $t$-test at the 95% confidence level. Values statistically indistinguishable from the best performance are shown in bold. Our LSB-CMM model out-performs the other two methods on the BirdSong database, and its performance is comparable to SIM on the MSRCv2 dataset and to CLPL on the Lost dataset. It should be noted that the input features are very coarse, which means that the cluster structure of the data is not well maintained. The relatively low performance of the SVM confirms this. If the instances were more precisely described by finer features, one would expect our model to perform better in those cases as well.

## 4    Conclusions

This paper introduced the Logistic Stick-Breaking Conditional Multinomial Model to address the superset label learning problem. The mixture representation allows LSB-CMM to discover cluster structure that has predictive power for the superset labels in the training data. Hence, if two labels co-occur, LSB-CMM is not forced to choose one of them to assign to the training example but instead can create a region that maps to both of them. Nonetheless, each region does predict from a multinomial, so the model still ultimately seeks to predict a single label. Our experiments show that the performance of the model is either better than or comparable to state-of-the-art methods.

**Acknowledgment**

This material is based upon work supported by the National Science Foundation under Grant No. 1125228. The code as an R package is available at:
`http://web.engr.oregonstate.edu/˜liuli/files/LSB-CMM_1.0.tar.gz`.

## References

[1] C. M. Bishop. Pattern recognition and machine learning. Springer, 2006.

[2] F. Briggs & X. F. Fern & R. Raich. Rank-Loss Support Instance Machines for MIML Instance Annotation. In proc. KDD, 2012.

[3] C.-C. Chang & C.-J. Lin. LIBSVM: A Library for Support Vector Machines. ACM Trans. on Intelligent Systems and Technology, 2(3):1-27, 2011.

[4] T. Cour & B. Sapp & C. Jordan & B. Taskar. Learning From Ambiguously Labeled Images. In Proc. CVPR 2009.

[5] T. Cour & B. Sapp & B. Taskar. Learning from Partial Labels. Journal of Machine Learning Research, 12:1225-1261, 2011.

[6] A. Frank & A. Asuncion. UCI Machine Learning Repository [http://archive.ics.uci.edu/ml].

[7] Y. Grandvalet. Logistic Regression for Partial Labels. In Proc. IPMU, 2002.

[8] E. Hullermeier & J. Beringer. Learning from Ambiguously Labeled Examples. In Proc. IDA-05, 6th International Symposium on Intelligent Data Analysis Madrid, 2005.

[9] L. Jie & F. Orabona. Learning from Candidate Labeling Sets. In Proc. NIPS, 2010.

[10] R. Jin & Z. Ghahramani. Learning with Multiple Labels. In Proc. NIPS, 2002.

[11] L-P. Liu & T. Dietterich. A Conditional Multinomial Mixture Model for Superset Label Learning (Supplementary Materials),
`http://web.engr.oregonstate.edu/˜liuli/pdf/lsb_cmm_supp.pdf`.

[12] N. Nguyen & R. Caruana. Classification with Partial Labels. In Proc. KDD, 2008.

[13] L. Ren & L. Du & L. Carin & D. B. Dunson. Logistic Stick-Breaking Process. Journal of Machine Learning Research, 12:203-239, 2011.

[14] Y. W. Teh. Dirichlet Processes. Encyclopedia of Machine Learning, to appear. Springer.

[15] Z.-H. Zhou & M.-L. Zhang. Multi-Instance Multi-Label Learning with Application To Scene Classification. Advances in Neural Information Processing Systems, 19, 2007

[16] X. Zhu & A. B. Goldberg. Introduction to Semi-Supervised Learning. Synthesis Lectures on Artificial Intelligence and Machine Learning, 3(1):1-130, 2009.

